# The Efficiency and The Robustness of Natural Gradient Descent Learning Rule

**Howard Hua Yang**
Department of Computer Science
Oregon Graduate Institute
PO Box 91000, Portland, OR 97291, USA
hyang@cse.ogi.edu

**Shun-ichi Amari**
Lab. for Information Synthesis
RIKEN Brain Science Institute
Wako-shi, Saitama 351-01, JAPAN
amari@zoo.brain.riken.go.jp

## Abstract

The inverse of the Fisher information matrix is used in the natural gradient descent algorithm to train single-layer and multi-layer perceptrons. We have discovered a new scheme to represent the Fisher information matrix of a stochastic multi-layer perceptron. Based on this scheme, we have designed an algorithm to compute the natural gradient. When the input dimension $n$ is much larger than the number of hidden neurons, the complexity of this algorithm is of order $O(n)$. It is confirmed by simulations that the natural gradient descent learning rule is not only efficient but also robust.

## 1 INTRODUCTION

The inverse of the Fisher information matrix is required to find the Cramer-Rao lower bound to analyze the performance of an unbiased estimator. It is also needed in the natural gradient learning framework (Amari, 1997) to design statistically efficient algorithms for estimating parameters in general and for training neural networks in particular. In this paper, we assume a stochastic model for multi-layer perceptrons. Considering a Riemannian parameter space in which the Fisher information matrix is a metric tensor, we apply the natural gradient learning rule to train single-layer and multi-layer perceptrons. The main difficulty encountered is to compute the inverse of the Fisher information matrix of large dimensions when the input dimension is high. By exploring the structure of the Fisher information matrix and its inverse, we design a fast algorithm with lower complexity to implement the natural gradient learning algorithm.

## 2   A STOCHASTIC MULTI-LAYER PERCEPTRON

Assume the following model of a stochastic multi-layer perceptron:

$$z = \sum_{i=1}^{m} a_i \varphi(w_i^T x + b_i) + \xi \qquad (1)$$

where $(\cdot)^T$ denotes the transpose, $\xi \sim N(0, \sigma^2)$ is a Gaussian random variable, and $\varphi(x)$ is a differentiable output function for hidden neurons. Assume the multi-layer network has a n-dimensional input, m hidden neurons, a one dimensional output, and $m \leq n$. Denote $a = (a_1, \cdots, a_m)^T$ the weight vector of the output neuron, $w_i = (w_{1i}, \cdots, w_{ni})^T$ the weight vector of the i-th hidden neuron, and $b = (b_1, \cdots, b_m)^T$ the vector of thresholds for the hidden neurons. Let $W = [w_1, \cdots, w_m]$ be a matrix formed by column weight vectors $w_i$, then (1) can be rewritten as $z = a^T \varphi(W^T x + b) + \xi$. Here, the scalar function $\varphi$ operates on each component of the vector $W^T x + b$.

The joint probability density function (pdf) of the input and the output is

$$p(x, z; W, a, b) = p(z|x; W, a, b)p(x).$$

Define a loss function:

$$L(x, z; \theta) = -\log p(x, z; \theta) = l(z|x; \theta) - \log p(x)$$

where $\theta = (w_1^T, \cdots, w_m^T, a^T, b^T)^T$ includes all the parameters to be estimated and

$$l(z|x; \theta) = -\log p(z|x; \theta) = \frac{1}{2\sigma^2}(z - a^T \varphi(W^T x + b))^2.$$

Since $\frac{\partial L}{\partial \theta} = \frac{\partial l}{\partial \theta}$, the Fisher information matrix is defined by

$$G(\theta) = E[\frac{\partial L}{\partial \theta}(\frac{\partial L}{\partial \theta})^T] = E[\frac{\partial l}{\partial \theta}(\frac{\partial l}{\partial \theta})^T] \qquad (2)$$

The inverse of $G(\theta)$ is often used in the Cramer-Rao inequality:

$$E[\|\widehat{\theta} - \theta^*\|^2 \mid \theta^*] \geq \mathrm{Tr}(G^{-1}(\theta^*))$$

where $\widehat{\theta}$ is an unbiased estimator of a true parameter $\theta^*$.

For the on-line estimator $\widehat{\theta}_t$ based on the independent examples $\{(x_s, z_s), s = 1, \cdots, t\}$ drawn from the probability law $p(x, z; \theta^*)$, the Cramer-Rao inequality for the on-line estimator is

$$E[\|\widehat{\theta}_t - \theta^*\|^2 \mid \theta^*] \geq \frac{1}{t}\mathrm{Tr}(G^{-1}(\theta^*)) \qquad (3)$$

## 3   NATURAL GRADIENT LEARNING

Consider a parameter space $\Theta = \{\theta\}$ in which the divergence between two points $\theta_1$ and $\theta_2$ is given by the Kullback-Leibler divergence

$$D(\theta_1, \theta_2) = \mathrm{KL}[p(x, z; \theta_1)\|p(x, z; \theta_2)].$$

When the two points are infinitesimally close, we have the quadratic form

$$D(\theta, \theta + d\theta) = \frac{1}{2}d\theta^T G(\theta)d\theta. \qquad (4)$$

This is regarded as the square of the length of $d\theta$. Since $G(\theta)$ depends on $\theta$, the parameter space is regarded as a Riemannian space in which the local distance is defined by (4). Here, the Fisher information matrix $G(\theta)$ plays the role of the Riemannian metric tensor.

It is shown by Amari(1997) that the steepest descent direction of a loss function $C(\theta)$ in the Riemannian space $\Theta$ is

$$-\tilde{\nabla}C(\theta) = -G^{-1}(\theta)\nabla C(\theta).$$

The natural gradient descent method is to decrease the loss function by updating the parameter vector along this direction. By multiplying $G^{-1}(\theta)$, the covariant gradient $\nabla C(\theta)$ is converted into its contravariant form $G^{-1}(\theta)\nabla C(\theta)$ which is consistent with the contravariant differential form $dC(\theta)$.

Instead of using $l(z|x;\theta)$ we use the following loss function:

$$l_1(z|x;\theta) = \frac{1}{2}(z - a^T\varphi(W^Tx + b))^2.$$

We have proved in [5] that $G(\theta) = \frac{1}{\sigma^2}A(\theta)$ where $A(\theta)$ does not depend on the unknown $\sigma$. So $G^{-1}(\theta)\frac{\partial l}{\partial\theta} = A^{-1}(\theta)\frac{\partial l}{\partial\theta}$. The on-line learning algorithms based on the gradient $\frac{\partial l_1}{\partial\theta}$ and the natural gradient $A^{-1}(\theta)\frac{\partial l_1}{\partial\theta}$ are, respectively,

$$\theta_{t+1} = \theta_t - \frac{\mu}{t}\frac{\partial l_1}{\partial\theta}(z_t|x_t;\theta_t), \tag{5}$$

$$\theta_{t+1} = \theta_t - \frac{\mu'}{t}A^{-1}(\theta_t)\frac{\partial l_1}{\partial\theta}(z_t|x_t;\theta_t) \tag{6}$$

where $\mu$ and $\mu'$ are learning rates.

When the negative log-likelihood function is chosen as the loss function, the natural gradient descent algorithm (6) gives a Fisher efficient on-line estimator (Amari, 1997), i.e., the asymptotic variance of $\theta_t$ driven by (6) satisfies

$$E[(\theta_t - \theta^*)(\theta_t - \theta^*)^T \mid \theta^*] \approx \frac{1}{t}G^{-1}(\theta^*) \tag{7}$$

which gives the mean square error

$$E[\|\theta_t - \theta^*\|^2 \mid \theta^*] \approx \frac{1}{t}\text{Tr}(G^{-1}(\theta^*)). \tag{8}$$

The main difficulty in implementing the natural gradient descent algorithm (6) is to compute the natural gradient on-line. To overcome this difficulty, we studied the structure of the matrix $A(\theta)$ in [5] and proposed an efficient scheme to represent this matrix. Here, we briefly describe this scheme.

Let $A(\theta) = [A_{ij}]_{(m+2)\times(m+2)}$ be a partition of $A(\theta)$ corresponding to the partition of $\theta = (w_1^T, \cdots, w_m^T, a^T, b^T)^T$. Denote $u_i = w_i/\|w_i\|, i = 1, \cdots, m$, $U_1 = [u_1, \cdots, u_m]$ and $[v_1, \cdots, v_m] = U_1(U_1^TU_1)^{-1}$. It has been proved in [5] that those blocks in $A(\theta)$ are divided into three classes: $C_1 = \{A_{ij}, i, j = 1, \cdots, m\}$, $C_2 = \{A_{i,m+1}, A_{m+1,i}^T, A_{i,m+2}, A_{m+2,i}^T, i = 1, \cdots, m\}$ and $C_3 = \{A_{m+i,m+j}, i, j = 1, 2\}$. Each block in $C_1$ is a linear combination of matrices $u_kv_l^T, k, l = 1, \cdots, m$, and $\Omega_0 = I - \sum_{k=1}^m u_kv_k^T$. Each block in $C_2$ is a matrix whose column is a linear combination of $\{v_k, k = 1, \cdots, m.\}$. The coefficients in these combinations are integrals with respect to the multivariate Gaussian distribution $N(0, R_1)$ where

$R_1 = U_1^T U_1$ is $m \times m$. Each block in $C_3$ is an $m \times m$ matrix whose entries are also integrals with respect to $N(0, R_1)$. Detail expressions for these integrals are given in [5]. When $\varphi(x) = \text{erf}(\frac{x}{\sqrt{2}})$, using the techniques in (Saad and Solla, 1995), we can find the analytic expressions for most of these integrals.

The dimension of $A(\theta)$ is $(nm + 2m) \times (nm + 2m)$. When the input dimension $n$ is much larger than the number of hidden neurons, by using the above scheme, the space for storing this large matrix is reduced from $O(n^2)$ to $O(n)$. We also gave a fast algorithm in [5] to compute $A^{-1}(\theta)$ and the natural gradient with the time complexity $O(n^2)$ and $O(n)$ respectively. The trick is to make use of the structure of the matrix $A^{-1}(\theta)$.

## 4   SIMULATION

In this section, we give some simulation results to demonstrate that the natural gradient descent algorithm is efficient and robust .

### 4.1   Single-layer perceptron

Assume 7-dimensional inputs $x_t \sim N(0, I)$ and $\varphi(u) = \frac{1-e^{-u}}{1+e^{-u}}$. For the single-layer perceptron, $z = \varphi(w^T x)$, the on-line gradient descent (GD) and the natural GD algorithms are respectively

$$w_{t+1} = w_t + \mu_0(t)(z_t - \varphi(w_t^T x_t))\varphi'(w_t^T x_t)x_t \quad \text{and} \tag{9}$$

$$w_{t+1} = w_t + \mu_1(t)A^{-1}(w_t)(z_t - \varphi(w_t^T x_t))\varphi'(w_t^T x_t)x_t \tag{10}$$

where

$$A^{-1}(w) = \frac{1}{d_1(w)}I + \left(\frac{1}{d_2(w)} - \frac{1}{d_1(w)}\right)\frac{ww^T}{w^2}, \quad w = \|w\|, \tag{11}$$

$$d_1(w) = \frac{1}{\sqrt{2\pi}} \int_{-\infty}^{\infty} (\varphi'(wx))^2 e^{-\frac{x^2}{2}} dx > 0, \tag{12}$$

$$d_2(w) = \frac{1}{\sqrt{2\pi}} \int_{-\infty}^{\infty} (\varphi'(wx))^2 x^2 e^{-\frac{x^2}{2}} dx > 0, \tag{13}$$

and $\mu_0(t)$ and $\mu_1(t)$ are two learning rate schedules defined by $\mu_i(t) = \mu(\eta_i, c_i, \tau_i; t), i = 0, 1$. Here,

$$\mu(\eta, c, \tau; t) = \eta(1 + \frac{c}{\eta}\frac{t}{\tau})/(1 + \frac{c}{\eta}\frac{t}{\tau} + \frac{t^2}{\tau}). \tag{14}$$

is the search-then-converge schedule proposed by (Darken and Moody, 1992) . Note that $t < \tau$ is a "search phase" and $t > \tau$ is a "converge phase". When $\tau_i = 1$, the learning rate function $\mu_i(t)$ has no search phase but a weaker converge phase when $\eta_i$ is small. When $t$ is large, $\mu_i(t)$ decreases as $\frac{c_i}{t}$.

Randomly choose a 7-dimensional vector as $w^*$ for the teacher network:

$$w^* = [-1.1043, 0.4302, 1.1978, 1.5317, -2.2946, -0.7866, 0.4428]^T.$$

Choose $\eta_0 = 1.25$, $\eta_1 = 0.05$, $c_0 = 8.75$, $c_1 = 1$, and $\tau_0 = \tau_1 = 1$. These parameters are selected by trial and error to optimize the performance of the GD and the natural GD methods at the noise level $\sigma = 0.2$. The training examples $\{(x_t, z_t)\}$ are generated by $z_t = \varphi(w^{*T} x_t) + \xi_t$ where $\xi_t \sim N(0, \sigma^2)$ and $\sigma^2$ is unknown to the algorithms.

Let $w_t$ and $\widetilde{w}_t$ be the weight vectors driven by the equations (9) and (10) respectively. $\|w_t - w^*\|$ and $\|\widetilde{w}_t - w^*\|$ are error functions for the GD and the natural GD.

Denote $w^* = \|w^*\|$. From the equation (11), we obtain the Cramer-Rao Lower Bound (CRLB) for the deviation at the true weight vector $w^*$:

$$\mathrm{CRLB}(t) = \frac{\sigma}{\sqrt{t}}\sqrt{\frac{n-1}{d_1(w^*)} + \frac{1}{d_2(w^*)}}. \tag{15}$$

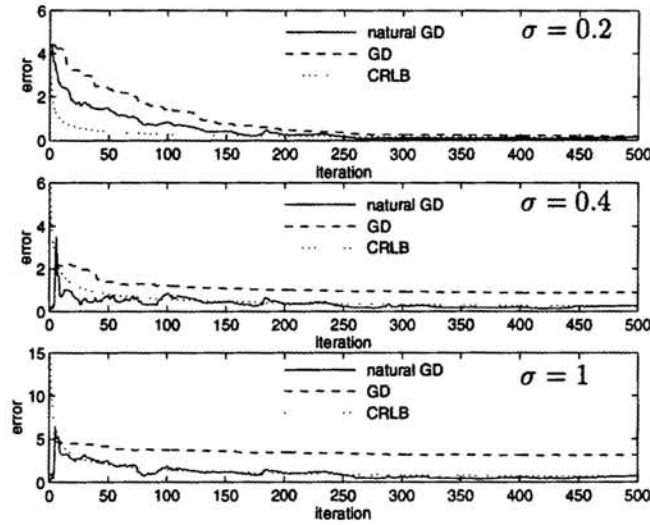

Figure 1: Performance of the GD and the natural GD at different noise levels $\sigma = 0.2, 0.4, 1$.

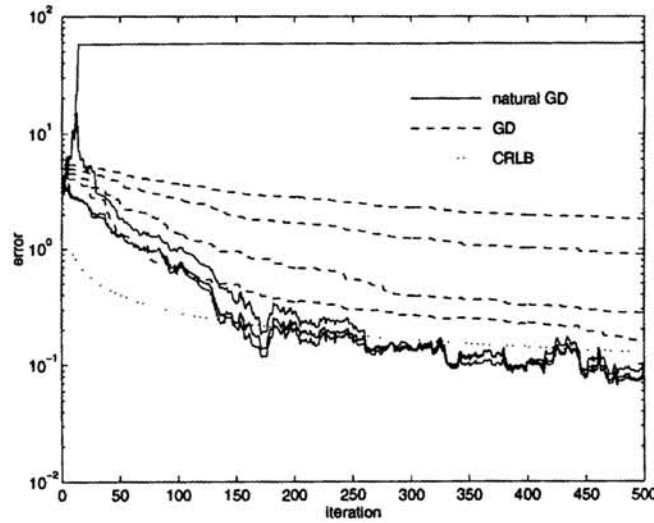

It is shown in Figure 1 that the natural GD algorithm reaches CRLB at different noise levels while the GD algorithm reaches the CRLB only at the noise level $\sigma = 0.2$. The robustness of the natural gradient descent against the additive noise in

Figure 2: Performance of the GD and the natural GD when $\eta_0 = 1.25, 1.75, 2.25, 2.75$, $\eta_1 = 0.05, 0.2, 0.4425, 0.443$, and $c_0 = 8.75$ and $c_1 = 1$ are fixed.

the training examples is clearly shown by Figure 1. When the teacher signal is non-stationary, our simulations show that the natural GD algorithm also reaches the CRLB.

Figure 2 shows that the natural GD algorithm is more robust than the GD algorithm against the change of the learning rate schedule. The performance of the GD algorithm deteriorates when the constant $\eta_0$ in the learning rate schedule $\mu_0(t)$ is different from that optimal one. On the contrary, the natural GD algorithm performs almost the same for all $\eta_1$ within a interval $[0.05, 0.4425]$. Figure 2 also shows that the natural GD algorithm breaks down when $\eta_1$ is larger than the critical number 0.443. This means that the weak converge phase in the learning rate schedule is necessary.

## 4.2 Multi-layer perceptron

Let us consider the simple multi-layer perceptron with 2-dimensional input and 2-hidden neurons. The problem is to train the committee machine $y = \varphi(w_1^T x) + \varphi(w_2^T x)$ based on the examples $\{(x_t, z_t), t = 1, \cdots, T\}$ generated by the stochastic committee machine $z_t = \varphi(w_1^{*T} x_t) + \varphi(w_2^{*T} x_t) + \xi_t$. Assume $\|w_i^*\| = 1$. We can reparameterize the weight vector to decrease the dimension of the parameter space from 4 to 2:

$$w_i = \begin{bmatrix} \cos(\alpha_i) \\ \sin(\alpha_i) \end{bmatrix}, \quad w_i^* = \begin{bmatrix} \cos(\alpha_i^*) \\ \sin(\alpha_i^*) \end{bmatrix}, \quad i = 1, 2.$$

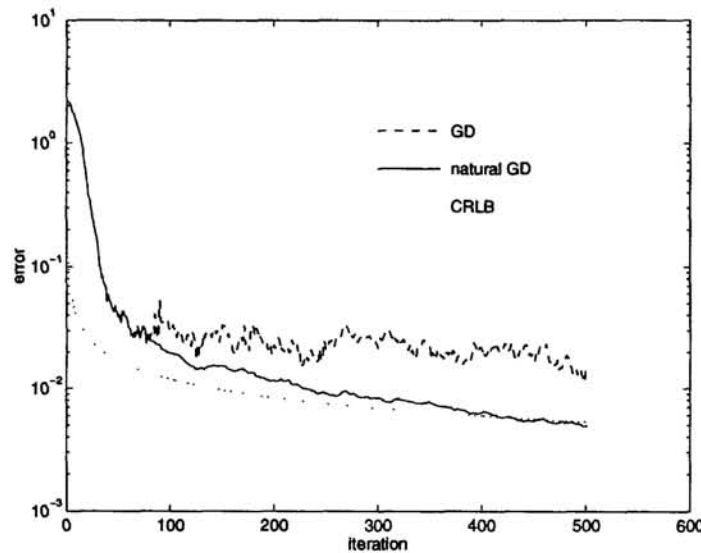

Figure 3: The GD vs. the natural GD

The parameter space is $\{\theta = (\alpha_1, \alpha_2)\}$. Assume that the true parameters are $\alpha_1^* = 0$ and $\alpha_2^* = \frac{3\pi}{4}$. Due to the symmetry, both $\theta_1^* = (0, \frac{3\pi}{4})$ and $\theta_2^* = (\frac{3\pi}{4}, 0)$ are true parameters. Let $\theta_t$ and $\theta_t'$ be computed by the GD algorithm and the natural GD

algorithm respectively. The errors are measured by

$$\varepsilon_t = \min\{\|\boldsymbol{\theta}_t - \boldsymbol{\theta}_1^*\|, \|\boldsymbol{\theta}_t - \boldsymbol{\theta}_2^*\|\}, \quad \text{and} \quad \varepsilon_t' = \min\{\|\boldsymbol{\theta}_t' - \boldsymbol{\theta}_1^*\|, \|\boldsymbol{\theta}_t' - \boldsymbol{\theta}_2^*\|\}.$$

In this simulation, using $\boldsymbol{\theta}_0 = (0.1, 0.2)$ as an initial estimate, we first start the GD algorithm and run it for 80 iterations. Then, we use the estimate obtained from the GD algorithm at the 80-th iteration as an initial estimate for the natural GD algorithm and run the latter algorithm for 420 iterations. The noise level is $\sigma = 0.05$. $N$ independent runs are conducted to obtain the errors $\varepsilon_t(j)$ and $\varepsilon_t'(j)$, $j = 1, \cdots, N$. Define root mean square errors

$$\bar{\varepsilon}_t = \sqrt{\frac{1}{N} \sum_{j=1}^{N} (\varepsilon_t(j))^2}, \quad \text{and} \quad \overline{\varepsilon'}_t = \sqrt{\frac{1}{N} \sum_{j=1}^{N} (\varepsilon_t'(j))^2}.$$

Based on $N = 10$ independent runs, the errors $\bar{\varepsilon}_t$ and $\overline{\varepsilon'}_t$ are computed and compared with the CRLB in Figure 3. The search-then-converge learning schedule (14) is used in the GD algorithm while the learning rate for the natural GD algorithm is simply the annealing rate $\frac{1}{k}$.

## 5 CONCLUSIONS

The natural gradient descent learning rule is statistically efficient. It can be used to train any adaptive system. But the complexity of this learning rule depends on the architecture of the learning machine. The main difficulty in implementing this learning rule is to compute the inverse of the Fisher information matrix of large dimensions. For a multi-layer perceptron, we have shown an efficient scheme to represent the Fisher information matrix based on which the space for storing this large matrix is reduced from $O(n^2)$ to $O(n)$. We have also shown an algorithm to compute the natural gradient. Taking advantage of the structure of the inverse of the Fisher information matrix, we found that the complexity of computing the natural gradient is $O(n)$ when the input dimension $n$ is much larger than the number of hidden neurons.

The simulation results have confirmed the fast convergence and statistical efficiency of the natural gradient descent learning rule. They have also verified that this learning rule is robust against the changes of the noise levels in the training examples and the parameters in the learning rate schedules.

## References

[1] S. Amari. Natural gradient works efficiently in learning. *Accepted by Neural Computation*, 1997.

[2] S. Amari. Neural learning in structured parameter spaces – natural Riemannian gradient. In *Advances in Neural Information Processing Systems, 9, ed. M. C. Mozer, M. I. Jordan and T. Petsche, The MIT Press: Cambridge, MA.*, pages 127–133, 1997.

[3] C. Darken and J. Moody. Towards faster stochastic gradient search. In *Advances in Neural Information Processing Systems, 4, eds. Moody, Hanson, and Lippmann, Morgan Kaufmann, San Mateo*, pages 1009–1016, 1992.

[4] D. Saad and S. A. Solla. On-line learning in soft committee machines. *Physical Review E*, 52:4225–4243, 1995.

[5] H. H. Yang and S. Amari. Natural gradient descent for training multi-layer perceptrons. *Submitted to IEEE Tr. on Neural Networks*, 1997.